# Neuronal Group Selection Theory: A Grounding in Robotics

**Jim Donnett and Tim Smithers**
Department of Artificial Intelligence
University of Edinburgh
5 Forrest Hill
Edinburgh  EH1 2QL
SCOTLAND

## ABSTRACT

In this paper, we discuss a current attempt at applying the organizational principle Edelman calls Neuronal Group Selection to the control of a real, two-link robotic manipulator. We begin by motivating the need for an alternative to the position-control paradigm of classical robotics, and suggest that a possible avenue is to look at the primitive animal limb 'neurologically ballistic' control mode. We have been considering a selectionist approach to coordinating a simple perception-action task.

## 1   MOTIVATION

The majority of industrial robots in the world are mechanical manipulators — often arm-like devices consisting of some number of rigid links with actuators mounted where the links join that move adjacent links relative to each other, rotationally or translationally.  At the joints there are typically also sensors measuring the relative *position* of adjacent links, and it is in terms of *position* that manipulators are generally controlled (a desired motion is specified as a desired position of the end effector, from which can be derived the necessary positions of the links comprising the manipulator). Position control dominates largely for historical reasons, rooted in *bang-bang control*: manipulators bumped between mechanical stops placed so as to enforce a desired trajectory for the end effector.

## 1.1   SERVOMECHANISMS

Mechanical stops have been superceded by position-controlling servomechanisms, negative feedback systems in which, for a typical manipulator with revolute joints, a desired joint angle is compared with a feedback signal from the joint sensor signalling actual measured angle; the difference controls the motive power output of the joint actuator proportionally.

Where a manipulator is constructed of a number of links, there might be a servomechanism for each joint. In combination, it is well known that joint motions can affect each other adversely, requiring careful design and analysis to reduce the possibility of unpleasant dynamical instabilities. This is especially important when the manipulator will be required to execute fast movements involving many or all of the joints. We are interested in such dynamic tasks, and acknowledge some successful servomechanistic solutions (see [Andersson 1988], who describes a ping pong playing robot), but seek an alternative that is not as computationally expensive.

## 1.2   ESCAPING POSITION CONTROL

In Nature, fast reaching and striking is a primitive and fundamental mode of control. In fast, time-optimal, neurologically ballistic movements (such as horizontal rotations of the head where subjects are instructed to turn it as fast as possible, [Hannaford and Stark 1985]), muscle activity patterns seem to show three phases: a launching phase (a burst of agonist), a braking phase (an antagonist burst), and a locking phase (a second agonist burst). Experiments have shown (see [Wadman *et al.* 1979]) that at least the first 100 mS of activity is the same even if a movement is blocked mechanically (without forewarning the subject), suggesting that the launch is specified from predetermined initial conditions (and is not immediately modified from proprioceptive information). With the braking and locking phases acting as a damping device at the end of the motion, the complete motion of the arm is essentially specified by the initial conditions — a mode radically differing from traditional robot positional control. The overall coordination of movements might even seem naive and simple when compared with the intricacies of servomechanisms (see [Braitenberg 1989, Nahvi and Hashemi 1984] who discuss the crane driver's strategy for shifting loads quickly and time-optimally).

The concept of letting insights (such as these) that can be gained from the biological sciences shape the engineering principles used to create artificial autonomous systems is finding favour with a growing number of researchers in robotics. As it is not generally trivial to see how life's devices can be mapped onto machines, there is a need for some fundamental experimental work to develop and test the basic theoretical and empirical components of this approach, and we have been considering various robotics problems from this perspective.

Here, we discuss an experimental two-link manipulator that performs a simple manipulation task — hitting a simple object perceived to be within its reach. The perception of the object specifies the initial conditions that determine an arm mo-

tion that reaches it. In relating initial conditions with motor currents, we have been considering a scheme based on Neuronal Group Selection Theory [Edelman 1987, Reeke and Edelman 1988], a theory of brain organization. We believe this to be the first attempt to apply selectionist ideas in a real machine, rather than just in simulation.

## 2   NEURONAL GROUP SELECTION THEORY

Edelman proposes Neuronal Group Selection (NGS) [Edelman 1978] as an organizing principle for higher brain function — mainly a biological basis for perception — primarily applicable to the mammalian (and specifically, human) nervous system [Edelman 1981]. The essential idea is that groups of cells, structurally varied as a result of developmental processes, comprise a population from which are selected those groups whose function leads to adaptive behaviour of the system. Similar notions appear in immunology and, of course, evolutionary theory, although the effects of neuronal group selection are manifest in the lifetime of the organism.

There are two premises on which the principle rests. The first is that the unit of selection is a cell group of perhaps 50 to 10,000 neurons. Intra-group connections between cells are assumed to vary (greatly) between groups, but other connections in the brain (particularly inter-group) are quite specific. The second premise is that the kinds of nervous systems whose organization the principle addresses are able to adapt to circumstances not previously encountered by the organism or its species [Edelman 1978].

### 2.1   THREE CENTRAL TENETS

There are three important ideas in the NGS theory [Edelman 1987].

- A first selective process (cell division, migration, differentiation, or death) results in structural diversity providing a *primary repertoire* of variant cell groups.

- A second selective process occurs as the organism experiences its environment; group activity that correlates with adaptive behaviour leads to differential amplification of intra- and inter-group synaptic strengths (the connectivity pattern remains unchanged). From the primary repertoire are thus selected groups whose adaptive functioning means they are more likely to find future use — these groups form the *secondary repertoire*.

- Secondary repertoires themselves form populations, and the NGS theory additionally requires a notion of *reentry*, or connections between repertoires, usually arranged in maps, of which the well-known retinotopic mapping of the visual system is typical. These connections are critical for they correlate motor and sensory repertoires, and lend the world the kind of spatiotemporal continuity we all experience.

## 2.2   REQUIREMENTS OF SELECTIVE SYSTEMS

To be selective, a system must satisfy three requirements [Reeke and Edelman 1988]. Given a configuration of input signals (ultimately from the sensory epithelia, but for 'deeper' repertoires mainly coming from other neuronal groups), if a group responds in a specific way it has *matched* the input [Edelman 1978]. The first requirement of a selective system is that it have a sufficiently large repertoire of variant elements to ensure that an adequate match can be found for a wide range of inputs. Secondly, enough of the groups in a repertoire must 'see' the diverse input signals effectively and quickly so that selection can operate on these groups. And finally, there must be a means for 'amplifying' the contribution, to the repertoire, of groups whose operation when matching input signals has led to adaptive behaviour.

In determining the necessary number of groups in a repertoire, one must consider the relationship between repertoire size and the specificity of member groups. On the one hand, if groups are very specific, repertoires will need to be very large in order to recognize a wide range of possible inputs. On the other hand, if groups are not as discriminating, it will be possible to have smaller numbers of them, but in the limit (a single group with virtually no specificity) different signals will no longer be distinguishable. A simple way to quantify this might be to assume that each of $N$ groups has a fixed probability, $p$, of matching an input configuration; then a typical measure [Edelman 1978] relating the effectiveness of recognition, $r$, to the number of groups is $r = 1 - (1 - p)^N$ (see Fig. 1).

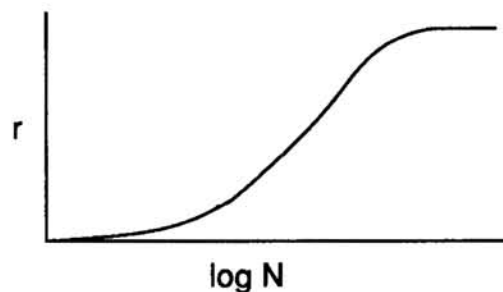

**Figure 1:** Recognition as a Function of Repertoire Size

From the shape of the curve in Fig. 1, it is clear that, for such a measure, below some lower threshold for $N$, the efficacy of recognition is equally poor. Similarly, above an upper threshold for $N$, recognition does not improve substantially as more groups are added.

## 3   SELECTIONISM IN OUR EXPERIMENT

Our manipulator is required to touch an object perceived to be within reach. This is a well-defined but non-trivial problem in motor-sensory coordination. Churchland proposes a geometrical solution for his two-eyed 'crab' [Churchland 1986], in which

eye angles are mapped to those joint angles (the crab has a two-link arm) that would bring the end of the arm to the point currently foveated by the eyes. Such a novel solution, in which computation is implicit and massively parallel, would be welcome; however, the crab is a simulation, and no heed is paid to the question of how the appropriate sensory-motor mapping could be generated for a real arm.

Reeke and Edelman discuss an automaton, Darwin III, similar to the crab, but which by selectional processes develops the ability to manipulate objects presented to it in its environment [Reeke and Edelman 1988]. The Darwin III simulation does not account for arm dynamics; however, Edelman suggests that the training paradigm is able to handle dynamic effects as well as the geometry of the problem [Edelman 1989]. We are attempting to implement a mechanical analogue of Darwin III, somewhat simplified, but which will experience the real dynamics of motion.

## 3.1   EXPERIMENTAL ARCHITECTURE AND HARDWARE

The mechanical arrangement of our manipulator is shown in Fig. 2. The two links have agonist/antagonist tendon-drive arrangement, with an actuator per tendon. There are strain gauges in-line with the tendons. A manipulator 'reach' is specified by six parameters: burst amplitude and period for each of the three phases, launch, brake, and lock.

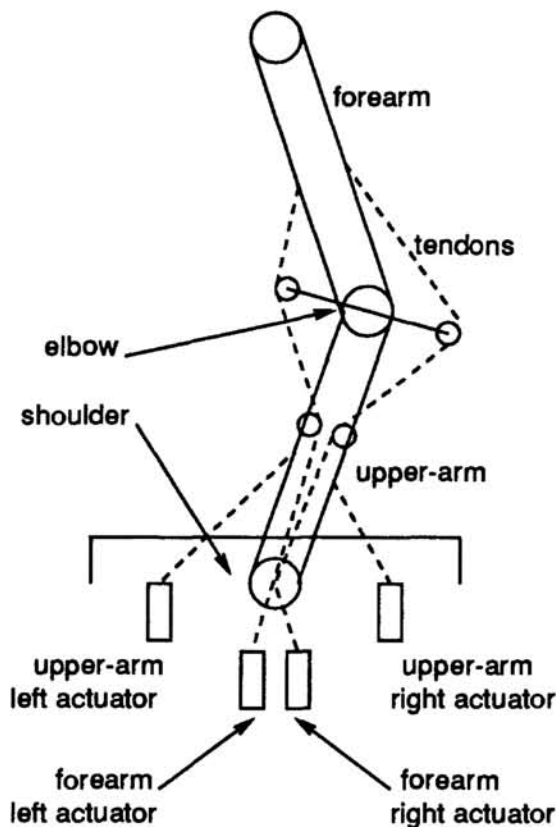

**Figure 2:** Manipulator Mechanical Configuration

At the end of the manipulator is an array of eleven pyroelectic-effect infrared detectors arranged in a U-shaped pattern. The *relative* location of a warm object presented to the arm is registered by the sensors, and is converted to eleven 8-bit integers. Since the sensor output is proportional to detected infrared energy flux, objects at the same temperature will give a more positive reading if they are close to the sensors than if they are further away. Also, a near object will register on adjacent sensors, not just on the one oriented towards it. Therefore, for a single, small object, a histogram of the eleven values will have a peak, and showing two things (Fig. 3): the sensor 'seeing' the most flux indicates the relative direction of the object, and the sharpness of the peak is proportional to the distance of the object.

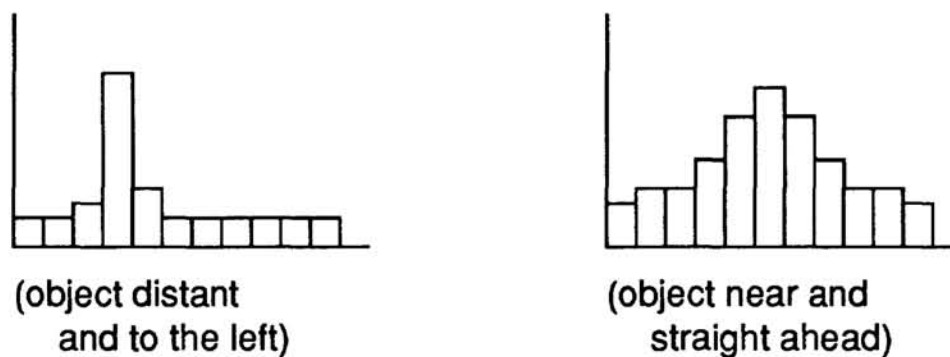

**Figure 3:** Histograms for Distant Versus Near Objects

Modelled on Darwin III [Reeke and Edelman 1988], the architecture of the selectional perception-action coordinator is as in Fig. 4. The boxes represent repertoires of appropriately interconnected groups of 'neurons'.

Darwin III responds mainly to contour in a two-dimensional world, analogous to the recognition of histogram shape in our system. Where Darwin III's 'unique response' network is sensitive to line segment lengths and orientations, ours is sensitive to the length of subsequences in the array of sensor output values in which values increase or decrease by the same amount, and the amounts by which they change; similarly, where Darwin III's 'generic response' network is sensitive to presence of or *changes* in orientation of lines, ours responds to the presence of the subsequences mentioned above, and the positions in the array where two subsequences abut.

The recognition repertoires are reciprocally connected, and both connect to the motor repertoire which consists of ballistic-movement 6-tuples. The system considers 'touching perceived object' to be adaptive, so when recognition activity correlates with a given 6-tuple, amplification ensures that the same response will be favoured in future.

# 4  WORK TO DATE

As the sensing system is not yet functional, this aspect of the system is currently simulated in an IBM PC/AT. The rest of the electrical and mechanical hardware is in place. The major difficulty currently faced is that the selectional system will become computationally intensive on a serial machine.

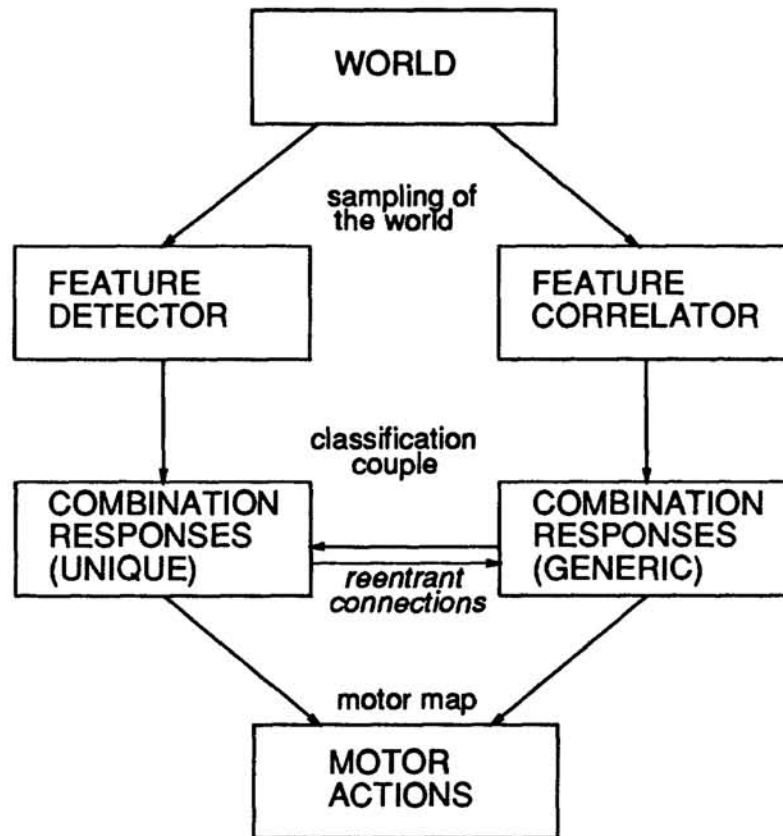

**Figure 4:** Experimental Architecture

For each possible ballistic 'reach', there must be a representation for the 'reach 6-tuple'. Therefore, the motor repertoire must become large as the dexterity of the manipulator is increased. Similarly, as the array of sensors is extended (resolution increased, or field of view widened), the classification repertoires must also grow. On a serial machine, polling the groups in the repertoires must be done one at a time, introducing a substantial delay between the registration of object and the actual touch, precluding the interception by the manipulator of fast moving objects. We are exploring possibilities for parallelizing the selectional process (and have for this reason constructed a network of processing elements), with the expectation that this will lead us closer to fast, dynamic manipulation, at minimal computational expense.

# References

Russell L. Andersson. *A Robot Ping-Pong Player: Experiment in Real-Time Intelligent Control*. MIT Press, Cambridge, MA, 1988.

Valentino Braitenberg. "Some types of movement", in C.G. Langton, ed., *Artificial Life*, pp. 555–565, Addison-Wesley, 1989.

Paul M. Churchland. "Some reductive strategies in cognitive neurobiology". Mind, **95**:279-309, 1986.

Jim Donnett and Tim Smithers. "Behaviour-based control of a two-link ballistic arm". Dept. of Artificial Intelligence, University of Edinburgh, Research Paper *RP 458*, 1990.

Gerald M. Edelman. "Group selection and phasic reentrant signalling: a theory of higher brain function", in G.M. Edelman and V.B. Mountcastle, eds., *The Mindful Brain*, pp. 51–100, MIT Press, Cambridge, MA, 1978.

Gerald M. Edelman. "Group selection as the basis for higher brain function", in F.O. Schmitt et al., eds., *Organization of the Cerebral Cortex*, pp. 535–563, MIT Press, Cambridge, MA, 1981.

Gerald M. Edelman. *Neural Darwinism: The Theory of Neuronal Group Selection*. Basic Books, New York, 1987.

Gerald M. Edelman. Personal correspondence, 1989.

Blake Hannaford and Lawrence Stark. "Roles of the elements of the triphasic control signal". Experimental Neurology, **90**:619–634, 1985.

M.J. Nahvi and M.R. Hashemi. "A synthetic motor control system; possible parallels with transformations in cerebellar cortex", in J.R. Bloedel et al., eds., *Cerebellar Functions*, pp. 67–69, Springer-Verlag, 1984.

George N. Reeke Jr. and Gerald M. Edelman. "Real brains and artificial intelligence", in Stephen R. Graubard, ed., *The Artificial Intelligence Debate*, pp. 143–173, The MIT Press, Cambridge, MA, 1988.

W.J. Wadman, J.J. Denier van der Gon, R.H. Geuse, and C.R. Mol. "Control of fast goal-directed arm movements". Journal of Human Movement Studies, **5**:3–17, 1979.
